# Ranking with Large Margin Principle: Two Approaches*

**Amnon Shashua**
School of CS&E
Hebrew University of Jerusalem
Jerusalem 91904, Israel
*email:* shashua@cs.huji.ac.il

**Anat Levin**
School of CS&E
Hebrew University of Jerusalem
Jerusalem 91904, Israel
*email:* alevin@cs.huji.ac.il

## Abstract

We discuss the problem of ranking $k$ instances with the use of a "large margin" principle. We introduce two main approaches: the first is the "fixed margin" policy in which the margin of the closest neighboring classes is being maximized — which turns out to be a direct generalization of SVM to ranking learning. The second approach allows for $k - 1$ different margins where the sum of margins is maximized. This approach is shown to reduce to $\nu$-SVM when the number of classes $k = 2$. Both approaches are optimal in size of $2l$ where $l$ is the total number of training examples. Experiments performed on visual classification and "collaborative filtering" show that both approaches outperform existing ordinal regression algorithms applied for ranking and multi-class SVM applied to general multi-class classification.

## 1  Introduction

In this paper we investigate the problem of inductive learning from the point of view of predicting variables of ordinal scale [3, 7, 5], a setting referred to as *ranking learning* or *ordinal regression*. We consider the problem of applying the large margin principle used in Support Vector methods [12, 1] to the ordinal regression problem while maintaining an (optimal) problem size linear in the number of training examples.

Let $\mathbf{x}_i^j$ be the set of training examples where $j = 1, ..., k$ denotes the class number, and $i = 1, ..., i_j$ is the index within each class. Let $l = \sum_j i_j$ be the total number of training examples. A straight-forward generalization of the 2-class separating hyperplane problem, where a single hyperplane determines the classification rule, is to define $k - 1$ separating hyperplanes which would separate the training data into $k$ ordered classes by modeling the ranks as intervals on the real line — an idea whose origins are with the classical cumulative model [9], see also [7, 5]. The geometric interpretation of this approach is to look for $k - 1$ parallel hyperplanes represented by vector $\mathbf{w} \in R^n$ (the dimension of the input vectors) and scalars $b_1 \leq ... \leq b_{k-1}$ defining the hyperplanes $(\mathbf{w}, b_1), ..., (\mathbf{w}, b_{k-1})$, such that the

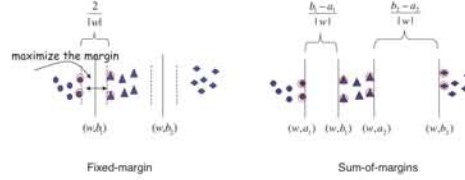

Figure 1: *Lefthand display*: fixed-margin policy for ranking learning. The margin to be maximized is associated with the two closest neighboring classes. As in conventional SVM, the margin is pre-scaled to be equal to $2/|\mathbf{w}|$ thus maximizing the margin is achieved by minimizing $\mathbf{w} \cdot \mathbf{w}$. The support vectors lie on the boundaries between the two closest classes. *Righthand display*: sum-of-margins policy for ranking learning. The objective is to maximize the sum of $k-1$ margins. Each class is sandwiched between two hyperplanes, the norm of $\mathbf{w}$ is set to unity as a constraint in the optimization problem and as a result the objective is to maximize $\sum_j (b_j - a_j)$. In this case, the support vectors lie on the boundaries among all neighboring classes (unlike the fixed-margin policy). When the number of classes $k=2$, the dual functional is equivalent to $\nu$-SVM.

data are *separated* by dividing the space into equally ranked regions by the decision rule

$$f(\mathbf{x}) = \min_{r \in \{1,...,k\}} \{r : \mathbf{w} \cdot \mathbf{x} - b_r < 0\}. \tag{1}$$

In other words, all input vectors $\mathbf{x}$ satisfying $b_{r-1} < \mathbf{w} \cdot \mathbf{x} < b_r$ are assigned the rank $r$ (using the convention that $b_k = \infty$). For instance, recently [5] proposed an "on-line" algorithm (with similar principles to the classic "perceptron" used for 2-class separation) for finding the set of parallel hyperplanes which would comply with the separation rule above.

To continue the analogy to 2-class learning, in addition to the separability constraints on the variables $\alpha = \{\mathbf{w}, b_1 \le ... \le b_{k-1}\}$ one would like to control the tradeoff between lowering the "empirical risk" $R_{emp}(\alpha)$ (error measure on the training set) and lowering the "confidence interval" $\Phi(\alpha, h)$ controlled by the VC-dimension $h$ of the set of loss functions. The "structural risk minimization" (SRM) principle [12] minimizes a bound on the risk over a structure on the set of functions. The geometric interpretation for 2-class learning is to *maximize* the margin between the boundaries of the two sets [12, 1].

In our setting of ranking learning, there are $k-1$ margins to consider, thus there are two possible approaches to take on the "large margin" principle for ranking learning:

"fixed margin" strategy: the margin to be maximized is the one defined by the *closest* (neighboring) pair of classes. Formally, let $\mathbf{w}, b_q$ be the hyperplane separating the two pairs of classes which are the closest among all the neighboring pairs of classes. Let $\mathbf{w}, b_q$ be scaled such the distance of the boundary points from the hyperplane is 1, i.e., the margin between the classes $q, q+1$ is $2/|\mathbf{w}|$ (see Fig. 1, lefthand display). Thus, the fixed margin policy for ranking learning is to find the direction $\mathbf{w}$ and the scalars $b_1, ..., b_{k-1}$ such that $\mathbf{w} \cdot \mathbf{w}$ is minimized (i.e., the margin between classes $q, q+1$ is maximized) subject to the separability constraints (modulo margin errors in the non-separable case).

"sum of margins" strategy: the sum of all $k-1$ margins are to be maximized. In this case, the margins are not necessarily equal (see Fig. 1, righthand display). Formally, the ranking

rule employs a vector $\mathbf{w}$, $|\mathbf{w}| = 1$, and a set of $2(k-1)$ thresholds $a_1 \leq b_1 \leq a_2 \leq b_2 \leq \ldots \leq a_{k-1} \leq b_{k-1}$ such that $\mathbf{w} \cdot \mathbf{x}_i^j \leq a_j$ and $\mathbf{w} \cdot \mathbf{x}_i^{j+1} \geq b_j$ for $j = 1, ..., k-1$. In other words, all the examples of class $1 \leq j \leq k$ are "sandwiched" between two parallel hyperplanes $(\mathbf{w}, a_j)$ and $(\mathbf{w}, b_{j-1})$, where $b_0 = -\infty$ and $a_k = \infty$. The $k-1$ margins are therefore $(b_j - a_j)$ and the large margin principle is to maximize $\sum_j (b_j - a_j)$ subject to the separability constraints above.

It is also fairly straightforward to apply the SRM principle and derive the bounds on the actual risk functional — see [11] for details.

In the remainder of this paper we will introduce the algorithmic implications of these two strategies for implementing the large margin principle for ranking learning. The fixed-margin principle will turn out to be a direct generalization of the Support Vector Machine (SVM) algorithm — in the sense that substituting $k = 2$ in our proposed algorithm would produce the dual functional underlying conventional SVM. It is interesting to note that the sum-of-margins principle reduces to $\nu$-SVM (introduced by [10] and later [2]) when $k = 2$.

## 2 Fixed Margin Strategy

Recall that in the fixed margin policy $(\mathbf{w}, b_q)$ is a "canonical" hyperplane normalized such that the margin between the closest classes $q, q+1$ is $2/\|\mathbf{w}\|$. The index $q$ is of course unknown. The unknown variables $\mathbf{w}, b_1 \leq \ldots \leq b_{k-1}$ (and the index $q$) could be solved in a two-stage optimization problem: a Quadratic Linear Programming (QLP) formulation followed by a Linear Programming (LP) formulation.

The (primal) QLP formulation of the ("soft margin") fixed-margin policy for ranking learning takes the form:

$$\min_{w,b_j,\epsilon_i^j,\epsilon_i^{*j+1}} \quad \frac{1}{2}\mathbf{w} \cdot \mathbf{w} + C \sum_i \sum_j \left( \epsilon_i^j + \epsilon_i^{*j+1} \right) \tag{2}$$

$$subject \ to$$

$$\mathbf{w} \cdot \mathbf{x}_i^j - b_j \leq -1 + \epsilon_i^j, \tag{3}$$

$$\mathbf{w} \cdot \mathbf{x}_i^{j+1} - b_j \geq 1 - \epsilon_i^{*j+1}, \tag{4}$$

$$\epsilon_i^j \geq 0, \epsilon_i^{*j} \geq 0 \tag{5}$$

where $j = 1, ..., k-1$ and $i = 1, ..., i_j$, and $C$ is some predefined constant. The scalars $\epsilon_i^j$ and $\epsilon_i^{*j+1}$ are positive for data points which are inside the margins or placed on the wrong side of the respective hyperplane. Since the margin is maximized while maintaining separability, it will be governed by the closest pair of classes because otherwise the separability conditions would cease to hold (modulo the choice of the constant $C$ which would tradeoff the margin size with possible margin errors — but that is discussed later).

The solution to this optimization problem is given by the saddle point of the Lagrange functional (Lagrangian):

$$
\begin{aligned}
L(\cdot) &= \frac{1}{2}\mathbf{w} \cdot \mathbf{w} + C \sum_{i,j} \left( \epsilon_i^j + \epsilon_i^{*j+1} \right) + \sum_{i,j} \lambda_i^j (\mathbf{w} \cdot \mathbf{x}_i^j - b_j + 1 - \epsilon_i^j) \\
&+ \sum_{i,j} \delta_i^j (1 - \epsilon_i^{*j+1} + b_j - \mathbf{w} \cdot \mathbf{x}_i^{j+1}) - \sum_{i,j} \zeta_i^j \epsilon_i^j - \sum_{i,j} \zeta_i^{*j+1} \epsilon_i^{*j+1}
\end{aligned}
$$

where $j = 1, ..., k-1, i = 1, ..., i_j$, and $\zeta_i^j, \zeta_i^{*j+1}, \lambda_i^j, \delta_i^j$ are all *non-negative* Lagrange multipliers. Since the primal problem is convex, there exists a strong duality between the primal and dual optimization functions. By first minimizing the Lagrangian with respect

to $\mathbf{w}, b_j, \epsilon_i^j, \epsilon_i^{*j+1}$ we obtain the dual optimization function which then must be maximized with respect to the Lagrange multipliers. From the minimization of the Lagrangian with respect to $\mathbf{w}$ we obtain:

$$\mathbf{w} = -\sum_{i,j} \lambda_i^j \mathbf{x}_i^j + \sum_{i,j} \delta_i^j \mathbf{x}_i^{j+1} \tag{6}$$

That is, the direction $\mathbf{w}$ of the parallel hyperplanes is described by a linear combination of the support vectors $\mathbf{x}$ associated with the non-vanishing Lagrange multipliers. From the Kuhn-Tucker theorem the support vectors are those vectors for which equality is achieved in the inequalities (3,4). These vectors lie on the two boundaries between the adjacent classes $q, q+1$ (and other adjacent classes which have the same margin). From the minimization of the Lagrangian with respect to $b_j$ we obtain the constraint:

$$\sum_i \lambda_i^j = \sum_i \delta_i^j \quad j = 1, ..., k-1 \tag{7}$$

and the minimization with respect to $\epsilon_i^j$ and $\epsilon_i^{*j+1}$ yields the constraints:

$$C - \lambda_i^j - \zeta_i^j = 0, \quad C - \delta_i^j - \zeta_i^{*j+1} = 0 \tag{8}$$

which in turn gives rise to the constraints $0 \leq \lambda_i^j \leq C$ where $\lambda_i^j = C$ if the corresponding data point is a margin error ($\zeta_i^j = 0$, thus from the Kuhn-Tucker theorem $\epsilon_i^j > 0$), and likewise for $\delta_i^j$. Note that a data point can count *twice* as a margin error — once with respect to the class on its "left" and once with respect to the class on its "right".

For the sake of presenting the dual functional in a compact form, we will introduce some new notations. Let $X^j$ be the $n \times i_j$ matrix whose columns are the data points $\mathbf{x}_i^j$, $i = 1, ..., i_j$. Let $\lambda^j = (\lambda_1^j, ..., \lambda_{i_j}^j)^\top$ be the vector whose components are the Lagrange multipliers $\lambda_i^j$ corresponding to class $j$. Likewise, let $\delta^j = (\delta_1^j, ..., \delta_{i_j}^j)^\top$ be the Lagrange multipliers $\delta_i^j$ corresponding to class $j + 1$. Let $\mu = (\lambda^1, ..., \lambda^{k-1}, \delta^1, ..., \delta^{k-1})^\top$ be the vector holding all the $\lambda_i^j$ and $\delta_i^j$ Lagrange multipliers, and let $\mu^1 = (\mu_1^1, ..., \mu_{k-1}^1)^\top = (\lambda^1, ..., \lambda^{k-1})^\top$ and $\mu^2 = (\mu_1^2, ..., \mu_{k-1}^2)^\top = (\delta^1, ..., \delta^{k-1})^\top$ the first and second halves of $\mu$. Note that $\mu_j^1 = \lambda^j$ is a vector, and likewise so is $\mu_j^2 = \delta^j$. Let $\mathbf{1}$ be the vector of 1's, and finally, let $Q$ be the matrix holding two copies of the training data:

$$Q = \left[ -X^1, ..., -X^{k-1}, X^2, ..., X^k \right]_{n \times N}, \tag{9}$$

where $N = 2l - i_1 - i_k$. For example, (6) becomes in the new notations $\mathbf{w} = Q\mu$. By substituting the expression for $\mathbf{w} = Q\mu$ back into the Lagrangian and taking into account the constraints (7,8) one obtains the dual functional which should be maximized with respect to the Lagrange multipliers $\mu_i$:

$$\max_\mu \quad \sum_{i=1}^N \mu_i - \mu^\top (Q^\top Q)\mu \tag{10}$$

$$subject\ to$$

$$0 \leq \mu_i \leq C \quad i = 1, ..., N \tag{11}$$

$$\mathbf{1} \cdot \mu_j^1 = \mathbf{1} \cdot \mu_j^2 \quad j = 1, ..., k-1 \tag{12}$$

Note that $k = 2$, i.e., we have only two classes thus the ranking learning problem is equivalent to the 2-class classification problem, the dual functional reduces and becomes equivalent to the dual form of conventional SVM. In that case $(Q^\top Q)_{ij} = y_i y_j \mathbf{x}_i \cdot \mathbf{x_j}$ where $y_i, y_j = \pm 1$ denoting the class membership.

Also worth noting is that since the dual functional is a function of the Lagrange multipliers $\lambda_i^j$ and $\delta_i^j$ alone, the problem size (the number of unknown variables) is equal to twice the number of training examples — precisely $N = 2l - i_1 - i_k$ where $l$ is the number of training examples. This favorably compares to the $O(l^2)$ required by the recent SVM approach to ordinal regression introduced in [7] or the $kl$ required by the general multi-class approach to SVM [4, 8].

Further note that since the entries of $Q^\top Q$ are the inner-products of the training examples, they can be represented by the kernel inner-product in the input space dimension rather than by inner-products in the feature space dimension. The decision rule, in this case, given a new instance vector $\mathbf{x}$ would be the rank $r$ corresponding to the first smallest threshold $b_r$ for which

$$\sum_{\substack{support\ vectors}} \delta_i^j K(\mathbf{x}_i^{j+1}, \mathbf{x}) - \sum_{\substack{support\ vectors}} \lambda_i^j K(\mathbf{x}_i^j, \mathbf{x}) \le b_r,$$

where $K(\mathbf{x}, \mathbf{y}) = \phi(\mathbf{x}) \cdot \phi(\mathbf{y})$ replaces the inner-products in the higher-dimensional "feature" space $\phi(\mathbf{x})$.

Finally, from the dual form one can solve for the Lagrange multipliers $\mu_i$ and in turn obtain $\mathbf{w} = Q\mu$ the direction of the parallel hyperplanes. The scalar $b_q$ (separating the adjacent classes $q, q + 1$ which are the closest apart) can be obtained from the support vectors, but the remaining scalars $b_j$ cannot. Therefore an additional stage is required which amounts to a Linear Programming problem on the original primal functional (2) but this time $\mathbf{w}$ is already known (thus making this a linear problem instead of a quadratic one).

## 3    Sum-of-Margins Strategy

In this section we propose an alternative large-margin policy which allows for $k - 1$ margins where the criteria function maximizes the *sum* of them. The challenge in formulating the appropriate optimization functional is that one cannot adopt the "pre-scaling" of $\mathbf{w}$ approach which is at the center of conventional SVM formulation and of the fixed-margin policy for ranking learning described in the previous section.

The approach we take is to represent the primal functional using $2(k - 1)$ parallel hyperplanes instead of $k - 1$. Each class would be "sandwiched" between two hyperplanes (except the first and last classes). Formally, we seek a ranking rule which employs a vector $\mathbf{w}$ and a set of $2(k - 1)$ thresholds $a_1 \le b_1 \le a_2 \le b_2 \le ... \le a_{k-1} \le b_{k-1}$ such that $\mathbf{w} \cdot \mathbf{x}_i^j \le a_j$ and $\mathbf{w} \cdot \mathbf{x}_i^{j+1} \ge b_j$ for $j = 1, ..., k - 1$. In other words, all the examples of class $1 \le j \le k$ are "sandwiched" between two parallel hyperplanes $(\mathbf{w}, a_j)$ and $(\mathbf{w}, b_{j-1})$, where $b_0 = -\infty$ and $a_k = \infty$.

The margin between two hyperplanes separating class $j$ and $j + 1$ is: $(b_j - a_j)/\sqrt{\|\mathbf{w}\|}$. Thus, by setting the magnitude of $\mathbf{w}$ to be of unit length (as a constraint in the optimization problem), the margin which we would like to maximize is $\sum_j (b_j - a_j)$ for $j = 1, ..., k - 1$ which we can formulate in the following *primal* QLP (see also Fig. 1, righthand display):

$$\min_{w, a_j, b_j} \quad \sum_{j=1}^{k-1} (a_j - b_j) + C \sum_i \sum_j \left( \epsilon_i^j + \epsilon_i^{*j+1} \right) \tag{13}$$

$$subject\ to$$

$$a_j \le b_j, \tag{14}$$

$$b_j \le a_{j+1}, \quad j = 1, ..., k - 2 \tag{15}$$

$$\mathbf{w} \cdot \mathbf{x}_i^j \le a_j + \epsilon_i^j, \quad b_j - \epsilon_i^{*j+1} \le \mathbf{w} \cdot \mathbf{x}_i^{j+1}, \tag{16}$$

$$\mathbf{w} \cdot \mathbf{w} \le 1, \quad \epsilon_i^j \ge 0, \epsilon_i^{*j+1} \ge 0 \tag{17}$$

where $j = 1, ..., k - 1$ (unless otherwise specified) and $i = 1, ..., i_j$, and $C$ is some predefined constant (whose physical role would be explained later). Note that the (non-convex) constraint $\mathbf{w} \cdot \mathbf{w} = 1$ is replaced by the convex constraint $\mathbf{w} \cdot \mathbf{w} \leq 1$ since it can be shown that the optimal solution $\mathbf{w}^*$ would have unit magnitude in order to optimize the objective function (see [11] for details). We will proceed to derive the dual functional below.

The Lagrangian takes the following form:

$$
\begin{aligned}
L(\cdot) &= \sum_j (a_j - b_j) + C \sum_{i,j} \left( \epsilon_i^j + \epsilon_i^{*j+1} \right) + \sum_j \psi_j (a_j - b_j) + \sum_{j=1}^{k-2} \eta_j (b_j - a_{j+1}) \\
&+ \sum_{i,j} \lambda_i^j (\mathbf{w} \cdot \mathbf{x}_i^j - a_j - \epsilon_i^j) + \sum_{i,j} \delta_i^j (b_j - \epsilon_i^{*j+1} - \mathbf{w} \cdot \mathbf{x}_i^{j+1}) \\
&+ \alpha (\mathbf{w} \cdot \mathbf{w} - 1) - \sum_{i,j} \zeta_i^j \epsilon_i^j - \sum_{i,j} \zeta_i^{*j+1} \epsilon_i^{*j}
\end{aligned}
$$

where $j = 1, ..., k - 1$ (unless otherwise specified), $i = 1, ..., i_j$, and $\psi_j, \eta_j, \alpha, \zeta_i^j, \zeta_i^{*j}, \lambda_i^j, \delta_i^j$ are all *non-negative* Lagrange multipliers. Due to lack of space we will omit further derivations (those can be found in [11]) and move directly to the dual functional which takes the following form:

$$
\max_{\boldsymbol{\mu}} \quad -\boldsymbol{\mu}^\top (Q^\top Q) \boldsymbol{\mu} \tag{18}
$$

$$
\textit{subject to}
$$

$$
0 \leq \mu_i \leq C \quad i = 1, ..., N \tag{19}
$$

$$
\mathbf{1} \cdot \mu_1^1 \geq 1, \quad \mathbf{1} \cdot \mu_{k-1}^2 \geq 1 \tag{20}
$$

$$
\mathbf{1} \cdot \mu^1 = \mathbf{1} \cdot \mu^2 \tag{21}
$$

where $Q$ and $\boldsymbol{\mu}$ are defined in the previous section. The direction $\mathbf{w}$ is represented by the linear combination of the support vectors: $\mathbf{w} = Q\boldsymbol{\mu}/\|Q\boldsymbol{\mu}\|$ where, following the Kuhn-Tucker theorem, $\mu_i > 0$ for all vectors on the boundaries between the adjacent pairs of classes and margin errors. In other words, the vectors $\mathbf{x}$ associated with non-vanishing $\mu_i$ are those which lie on the hyperplanes or vectors tagged as margin errors. Therefore, all the thresholds $a_j, b_j$ can be recovered from the support vectors — unlike the fixed-margin scheme which required another LP pass.

The dual functional (18) is similar to the dual functional (10) but with some crucial differences: (i) the quadratic criteria functional is homogeneous, and (ii) constraints (20) lead to the constraint $\sum_i \mu_i \geq 2$. These two differences are also what distinguishes between conventional SVM and $\nu$-SVM for 2-class learning proposed recently by [10]. Indeed, if we set $k = 2$ in the dual functional (18) we would be able to conclude that the two dual functionals are identical (by a suitable change of variables). Therefore, the role of the constant $C$ complies with the findings of [10] by controlling the tradeoff between the number of margin errors and support vectors and the size of the margins: $2/N \leq C \leq 2$ such that when $C = 2$ a single margin error is allowed (otherwise a duality gap would occur) and when $C = 2/N$ all vectors are allowed to become margin errors and support vectors (see [11] for a detailed discussion on this point).

In the general case of $k > 2$ classes (in the context of ranking learning) the role of the constant $C$ carries the same meaning: $C \leq 2(k-1)/\#\text{m.e.}$ where $\#\text{m.e.}$ stand for "total number of margin errors", thus

$$
\frac{2(k-1)}{N} \leq C \leq 2(k-1).
$$

Since a data point can can count twice for a margin error, the total number of margin errors in the worst case is $N = 2l - i_1 - i_k$ where $l$ is the total number of data points.

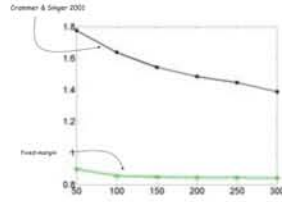

Figure 2: The results of the fixed-margin principle plotted against the results of PRank of [5] which does not use a large-margin principle. The average error of PRank is about 1.25 compared to 0.7 with the fixed-margin algorithm.

## 4 Experiments

Due to lack of space we describe only two sets of experiments we conducted on a "collaborative filtering" problem and visual data ranking. More details and further experiments are reported in [11].

In general, the goal in collaborative filtering is to predict a person's rating on new items such as movies given the person's past ratings on similar items and the ratings of other people of all the items (including the new item). The ratings are ordered, such as "highly recommended", "good",..., "very bad" thus collaborative filtering falls naturally under the domain of ordinal regression (rather than general multi-class learning).

The "EachMovie" dataset [6] contains 1628 movies rated by 72,916 people arranged as a 2D array whose columns represent the movies and the rows represent the users — about 5% of the entries of this array are filled-in with ratings between $0, ..., 6$ totaling 2,811,983 ratings. Given a new user, the ratings of the user on the 1628 movies (not all movies would be rated) form the $y_i$ and the i'th column of the array forms the $\mathbf{x}_i$ which together form the training data (for that particular user). Given a new movie represented by the vector $\mathbf{x}$ of ratings of all the other 72,916 users (not all the users rated the new movie), the learning task is to predict the rating $f(\mathbf{x})$ of the new user. Since the array contains empty entries, the ratings were shifted by $-3.5$ to have the possible ratings $\{-2.5, -1.5, -0.5, 0.5, 1.5, 2.5\}$ which allows to assign the value of zero to the empty entries of the array (movies which were not rated).

For the training phase we chose users which ranked about 450 movies and selected a subset $\{50, 100, ..., 300\}$ of those movies for training and tested the prediction on the remaining movies. We compared our results (collected over 100 runs) — the average distance between the correct rating and the predicted rating — to the best "on-line" algorithm of [5] called "PRank" (there is no use of large margin principle). In their work, PRank was compared to other known on-line approaches and was found to be superior, thus we limited our comparison to PRank alone. Attempts to compare our algorithms to other known ranking algorithms which use a large-margin principle ([7], for example) were not successful since those square the training set size which made the experiment with the Eachmovie dataset untractable computationally.

The graph in Fig. 2 shows that the large margin principle makes a significant difference on the results compared to PRank. The results we obtained with PRank are consistent with the reported results of [5] (best average error of about 1.25), whereas our fixed-margin algorithm provided an average error of about 0.7).

We have applied our algorithms to classification of "vehicle type" to one of three classes: "small" (passenger cars), "medium" (SUVs, minivans) and "large" (buses, trucks). There

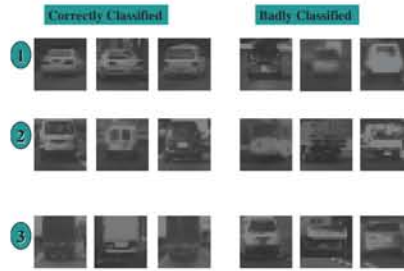

Figure 3: Classification of vehicle type: Small, Medium and Large (see text for details).

is a natural order Small, Medium, Large since making a mistake between Small and Large is worse than confusing Small and Medium, for example. We compared the classification error (counting the number of miss-classifications) to general multi-class learning using pair-wise SVM. The error over a test set of about 14,000 pictures was 20% compared to 25% when using general multi-class SVM. We also compared the error (averaging the difference between the true rank $\{1, 2, 3\}$ and the predicted rank using 2nd-order kernel) to PRank. The average error was 0.216 compared to 1.408 with PRank. Fig. 3 shows a typical collection of correctly classified and incorrectly classified pictures from the test set.

## Footnotes

*This work was done while A.S. was spending his sabbatical at the computer science department of Stanford University.

## References

[1] B.E. Boser, I.M. Guyon, and V.N. Vapnik. A training algorithm for optimal margin classifers. In *Proc. of the 5th ACM Workshop on Computational Learning Theory*, pages 144–152. ACM Press, 1992.

[2] C.C. Chang and C.J. Lin. Training $\nu$–Support Vector classifiers: Theory and Algorithms. In *Neural Computations*, 14(8), 2002.

[3] W.W. Cohen, R.E. Schapire, and Y. Singer. Learning to order things. *Journal of Artificial Intelligence Research (JAIR)*, 10:243–270, 1999.

[4] K. Crammer and Y. Singer. On the algorithmic implementation of multiclass kernel-based vector machines. *Journal of Machine Learning Research*, 2:265–292, 2001.

[5] K. Crammer and Y. Singer. Pranking with ranking. In *Proceedings of the conference on Neural Information Processing Systems (NIPS)*, 2001.

[6] http://www.research.compaq.com/SRC/eachmovie/.

[7] R. Herbrich, T. Graepel, and K. Obermayer. Large margin rank boundaries for ordinal regression. Advances in Large Margin Classifiers, 2000. pp. 115–132.

[8] Y. Lee, Y. Lin, and G. Wahba. Multicategory support vector machines. Technical Report 1043, Univ. of Wisconsin, Dept. of Statistics, Sep. 2001.

[9] P. McCullagh and J. A. Nelder. *Generalized Linear Models*. Chapman and Hall, London, 2nd edition edition, 1989.

[10] B. Scholkopf, A. Smola, R.C. Williamson, and P.L. Bartless. New support vector algorithms. *Neural Computation*, 12:1207–1245, 2000.

[11] A. Shashua and A. Levin. Taxonomy of Large Margin Principle Algorithms for Ordinal Regression Problems. Technical Report 2002-39, Leibniz Center for Research, School of Computer Science and Eng., the Hebrew University of Jerusalem.

[12] V.N. Vapnik. *The nature of statistical learning*. Springer, 2nd edition, 1998.

[13] J. Weston and C. Watkins. Support vector machines for multi-class pattern recognition. In *Proc. of the 7th European Symposium on Artificial Neural Networks*, April 1999.
